# Heterogeneous Component Analysis

**Shigeyuki Oba[1], Motoaki Kawanabe[2], Klaus Robert Müller[3,2], and Shin Ishii[4,1]**
1. Graduate School of Information Science, Nara Institute of Science and Technology, Japan
2. Fraunhofer FIRST.IDA, Germany
3. Department of Computer Science, Technical University Berlin, Germany
4. Graduate School of Informatics, Kyoto University, Japan
shige-o@is.naist.jp

## Abstract

In bioinformatics it is often desirable to combine data from various measurement sources and thus structured feature vectors are to be analyzed that possess different intrinsic blocking characteristics (e.g., different patterns of missing values, observation noise levels, effective intrinsic dimensionalities). We propose a new machine learning tool, heterogeneous component analysis (HCA), for feature extraction in order to better understand the factors that underlie such complex structured heterogeneous data. HCA is a linear block-wise sparse Bayesian PCA based not only on a probabilistic model with block-wise residual variance terms but also on a Bayesian treatment of a block-wise sparse factor-loading matrix. We study various algorithms that implement our HCA concept extracting sparse heterogeneous structure by obtaining common components for the blocks and specific components within each block. Simulations on toy and bioinformatics data underline the usefulness of the proposed structured matrix factorization concept.

## 1 Introduction

Microarray and other high-throughput measurement devices have been applied to examine specimens such as cancer tissues of biological and/or clinical interest. The next step is to go towards combinatorial studies in which tissues measured by two or more of such devices are simultaneously analyzed. However, such combinatorial studies inevitably suffer from differences in experimental conditions, or, even more complex, from different measurement technologies. Also, when concatenating a data set from different measurement sources, we often observe systematic missing parts in a dataset (e.g., Fig 3A). Moreover, the noise levels may vary among different experiments. All these induce a *heterogeneous* structure in data, that needs to be treated appropriately. Our work will contribute exactly to this topic, by proposing a Bayesian method for feature subspace extraction, called heterogeneous component analysis (HCA, sections 2 and 3). HCA performs a linear feature extraction based on matrix factorization in order to obtain a sparse and structured representation. After relating to previous methods (section 4), HCA is applied to toy data and more interestingly to neuroblastoma data from different measurement techniques (section 5). We obtain interesting factors that may be a first step towards better biological model building.

## 2 Formulation of the HCA problem

Let a matrix $\boldsymbol{Y} = \{y_{ij}\}_{i=1:M,j=1:N}$ denote a set of $N$ observations of $M$-dimensional feature vectors, where $y_{ij} \in \mathbb{R}$ is the $j$-th observation of the $i$-th feature. In a heterogeneous situation, we assume the $M$-dimensional feature vector is decomposed into $L$ disjoint blocks. Let $I^{(l)}$ denote a set of feature indices included in the $l$-th block, so that $I^{(1)} \cup \cdots \cup I^{(L)} = I$ and $I^{(l)} \cap I^{(l')} = \emptyset$ for $l \neq l'$.

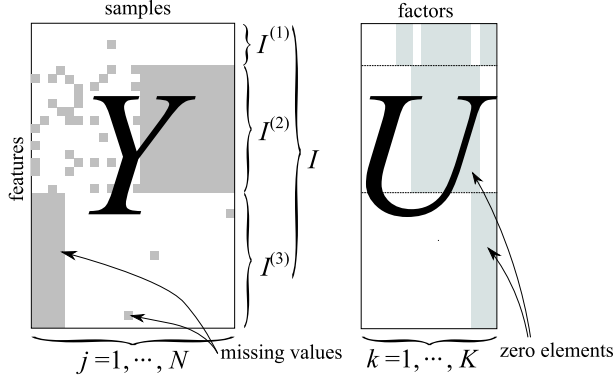

Figure 1: An illustration of a typical dataset and the result by the HCA. The observation matrix $\boldsymbol{Y}$ consists of multiple samples $j = 1, \ldots, N$ with high-dimensional features $i \in I$. The features consist of multiple blocks, in this case $I^{(1)} \cup I^{(2)} \cup I^{(3)} = I$. There are many missing observations whose distribution is highly structural depending on each block. HCA optimally factorizes the matrix $\boldsymbol{Y}$ so that the factor-loading matrix $\boldsymbol{U}$ has structural sparseness; it includes some regions of zero elements according to the block structure of the observed data. Each factor may or may not affect all the features within a block, but each block does not necessarily affect all the factors. Therefore, the rank of each factor-loading sub-matrix for each block (or any set of blocks) can be different from the others. The resulting block-wise sparse matrix reflects a characteristic *heterogeneity* of features over blocks.

We assume that the matrix $\boldsymbol{Y} \in \mathbb{R}^{M \times N}$ is a noisy observation of a matrix of true values $\boldsymbol{X} \in \mathbb{R}^{M \times N}$ whose rank is $K(< \min(M, N))$ and has a factorized form:

$$\boldsymbol{Y} = \boldsymbol{X} + \boldsymbol{E}, \boldsymbol{X} = \boldsymbol{U}\boldsymbol{V}^{\mathrm{T}}, \tag{1}$$

where $\boldsymbol{E} \in \mathbb{R}^{M \times N}$, $\boldsymbol{U} \in \mathbb{R}^{M \times K}$, and $\boldsymbol{V} \in \mathbb{R}^{N \times K}$ are matrices of residuals, factor-loadings, and factors, respectively. The superscript T denotes the matrix transpose. There may be missing or unmeasured observations denoted by a matrix $\boldsymbol{W} \in \{0, 1\}^{M \times N}$, which indicates observation $y_{ij}$ is missing if $w_{ij} = 0$ or exists otherwise ($w_{ij} = 1$).

Figure 1 illustrates the concept of HCA. In this example, the observed data matrix (left panel) is made up by three blocks of features. They have block-wise variation in effective dimensionalities, missing rates, observation noise levels, and so on, which we overall call *heterogeneity*. Such heterogeneity affects the effective rank of the observation sub-matrix corresponding to each block, and hence leads naturally to different ranks of factor-loading sub-matrix between blocks. In addition, there can exist block-wise patterns of missing values (shadowed rectangular regions in the left panel); such a situation would occur, for example in bioinformatics, when some particular genes have been measured in one assay (constituting a block) but not in another assay (constituting another block).

To better understand the objective data based on the feature extraction by matrix factorization, we assume a block-wise sparse factor-loading matrix $\boldsymbol{U}$ (right panel in Fig.1). Namely, the effective rank of an observation sub-matrix corresponding to a block is reflected by the number of non-zero components in the corresponding rows of $\boldsymbol{U}$. Assuming such a block-wise sparse structure can decrease the model's effective complexity, and will describe the data better and therefore lead to better generalization ability, e.g., for missing value prediction.

## 3   A probabilistic model for HCA

**Model**   For each element of the residual matrix, $e_{ij} \equiv y_{ij} - \sum_{k=1}^{K} u_{ik}v_{jk}$, we assume a Gaussian distribution with a common variance $\sigma_l^2$ for every feature $i$ in the same block $I^{(l)}$:

$$\ln p(e_{ij}|\sigma_{l(i)}^2) = -\frac{1}{2}\sigma_{l(i)}^{-2}e_{ij}^2 - \frac{1}{2}\ln \sigma_{l(i)}^2 - \frac{1}{2}\ln 2\pi, \tag{2}$$

where $l(i)$ denotes the pre-determined block index to which the $i$-th feature belongs. For a factor matrix $\boldsymbol{V}$, we assume a Gaussian prior:

$$\ln p(\boldsymbol{V}) = \sum_{j=1}^{N} \sum_{k=1}^{K} \left( -\frac{1}{2} v_{jk}^2 - \ln 2\pi \right). \tag{3}$$

The above two assumptions are exactly the same as those for probabilistic PCA that is a special case of HCA with a single active block. Another special case where each block contains only one active feature is probabilistic factor analysis (FA). Namely, maximum likelihood (ML) estimation based on the following log-likelihood includes both the PCA and the FA as special settings of the blocks.

$$\ln p(\boldsymbol{Y}, \boldsymbol{V} | \boldsymbol{U}, \boldsymbol{\sigma}^2) = \frac{1}{2} \sum_{ij} w_{ij} \left( -\sigma_{l(i)}^{-2} e_{ij}^2 - \ln \sigma_{l(i)}^2 - \ln 2\pi \right) + \frac{1}{2} \sum_{jk} \left( -v_{jk}^2 - \ln 2\pi \right). \tag{4}$$

$\boldsymbol{\sigma}^2 = (\sigma_l^2)_{l=1,\ldots,L}$ is a vector of variances of all blocks. Since $w_{ij} = 0$ iff $y_{ij}$ is missing, the summation $\sum_{ij}$ is actually taken for all observed values in $\boldsymbol{Y}$.

Another character of the HCA model is the block-wise sparse factor-loading matrix, which is implemented by a prior for $\boldsymbol{U}$, given by

$$\ln p(\boldsymbol{U}|\boldsymbol{T}) = \sum_{ik} t_{ik} \left( -\frac{1}{2} u_{ik}^2 - \frac{1}{2} \ln 2\pi \right), \tag{5}$$

where $\boldsymbol{T} = \{t_{ik}\}$ is a block-wise *mask matrix* which defines the block-wise-sparse structure; if $t_{ik} = 0$, then $u_{ik} = 0$ with probability 1. Each column vector of the mask matrix takes one of the possible block-wise mask patterns; a binary pattern vector whose dimensionality is the same as the factor-loading vector, and whose values are consistent, either 0 or 1, within each block. When there are $L$ blocks, each column vector of $\boldsymbol{T}$ can take one of $2^L$ possible patterns including the zero vector, and hence, the matrix $\boldsymbol{T}$ with $K$ columns can take one of $2^{LK}$ possible patterns.

**Parameter estimation** We estimated the model parameters $\boldsymbol{U}$ and $\boldsymbol{V}$ by maximum *a posteriori* (MAP) estimation, and $\boldsymbol{\sigma}$ by ML estimation; that is, the log-joint: $\mathcal{L} \stackrel{\text{def}}{=} \log P(\boldsymbol{Y}, \boldsymbol{U}, \boldsymbol{V} | \boldsymbol{\sigma}, \boldsymbol{T})$ was maximized w.r.t. $\boldsymbol{U}, \boldsymbol{V}$ and $\boldsymbol{\sigma}$.

Maximization of the log-joint $\mathcal{L}$ w.r.t $\boldsymbol{U}, \boldsymbol{V}$, and $\boldsymbol{\sigma}$ was performed by the conjugate gradient algorithm that was available in the NETLAB toolbox [1]. The stationary condition w.r.t. the variance, $\frac{\partial \mathcal{L}}{\partial (\boldsymbol{\sigma}^2)} = 0$, was solved as a closed form of $\boldsymbol{U}$ and $\boldsymbol{V}$:

$$\tilde{\sigma}_l^2(\boldsymbol{U}, \boldsymbol{V}) \stackrel{\text{def}}{=} \text{mean}_{(i,j|l)}[e_{ij}^2], \tag{6}$$

where $\text{mean}_{(i,j|l)}[.]$ is the average over all pairs $(i,j)$ not missing in the $l$-th block. By redefining the objective function with the closed form solution plugged in:

$$\tilde{\mathcal{L}}(\boldsymbol{U}, \boldsymbol{V}) \stackrel{\text{def}}{=} \mathcal{L}(\boldsymbol{U}, \boldsymbol{V}, \tilde{\boldsymbol{\sigma}}^2(\boldsymbol{U}, \boldsymbol{V})), \tag{7}$$

the conjugate gradient of $\tilde{\mathcal{L}}$ w.r.t. $\boldsymbol{U}$ and $\boldsymbol{V}$ led to faster and more stable optimization than the naive maximization of $\mathcal{L}$ w.r.t. $\boldsymbol{U}, \boldsymbol{V}$, and $\boldsymbol{\sigma}^2$.

**Model selection** The mask matrix $\boldsymbol{T}$ was determined by maximization of the log-marginal likelihood $\int \mathcal{L} d\boldsymbol{U} d\boldsymbol{V}$ which was calculated by Laplace approximation around the MAP estimator:

$$\mathcal{E}(\boldsymbol{T}) \stackrel{\text{def}}{=} \mathcal{L} - \frac{1}{2} \text{lndet} \boldsymbol{H}, \tag{8}$$

where $\boldsymbol{H} \stackrel{\text{def}}{=} \frac{\partial^2}{\partial \theta \partial \theta^{\mathrm{T}}} \mathcal{L}$ is the Hessian of log-joint w.r.t. all elements ($\theta$) in the parameters $\boldsymbol{U}$ and $\boldsymbol{V}$.

The log Hessian term, $\text{lndet} \boldsymbol{H}$, which works as a penalty term for maintaining non-zero elements in the factor-loading matrix, was simplified in order for tractable calculation. Namely, independence in the log-joint was assumed:

$$\frac{\partial^2 \mathcal{L}}{\partial u_{ik} v_{jk'}} \approx 0, \quad \frac{\partial^2 \mathcal{L}}{\partial u_{ik} u_{ik'}} \approx 0, \text{and} \quad \frac{\partial^2 \mathcal{L}}{\partial v_{jk} v_{jk'}} \approx 0, \tag{9}$$

which enabled a similar tractable computation to variational Bayes (VB) and was expected to produce satisfactory results.

To avoid searching through an exponentially large number of possibilities, we implemented a greedy search that optimizes each of the column vectors in a step-wise manner, called HCA-greedy algorithm. In each step of the HCA-greedy algorithm, factor-loading and factor vectors are estimated based on $2^L$ possible settings of block-wise mask vectors, and we accept the one achieving the maximum log-marginal. It terminated if zero vector is accepted as the best mask vector.

**HCA with ARD**  The greedy search still searches $2^L$ possibilities per a factor, whose computation increases exponentially as the number of blocks $L$ increases. The automatic relevance determination (ARD) is a hierarchical Bayesian approach for selecting relevant bases, which has been applied to component analyzers since its first introduction to Bayesian PCA (BPCA) [2].

The prior for $U$ is given by

$$\ln p(\boldsymbol{U}|\boldsymbol{\alpha}) = \frac{1}{2} \sum_{l=1}^{L} \sum_{k=1}^{K} \left\{ \sum_{i \in I_l} \left( -\alpha_{lk} u_{ik}^2 + \ln \alpha_{lk} - \ln 2\pi \right) \right\}, \tag{10}$$

where $\alpha_{lk}$ is an ARD hyper-parameter for the $l$-th block of the $k$-th column of $U$. $\boldsymbol{\alpha}$ is a vector of all elements of $\alpha_{lk}$, $l = 1, \ldots, L, k = 1, \ldots, K$. With this prior, the log-joint probability density function becomes

$$\ln p(\boldsymbol{Y}, \boldsymbol{U}, \boldsymbol{V}|\boldsymbol{\sigma}^2, \boldsymbol{\alpha}) = \frac{1}{2} \sum_{ij} w_{ij} \left( -\sigma_{l(i)}^{-2} e_{ij}^2 - \ln \sigma_{l(i)}^2 - \ln 2\pi \right) + \frac{1}{2} \sum_{jk} \left( -v_{jk}^2 - \ln 2\pi \right)$$

$$+ \frac{1}{2} \sum_{ik} \left( -\alpha_{l(i)k} u_{ik}^2 + \ln \alpha_{l(i)k} - \ln 2\pi \right). \tag{11}$$

According to this ARD approach, $\boldsymbol{\alpha}$ is updated by the conjugate gradient-based optimization simultaneously with $U$ and $V$. In each step of the optimization, $\boldsymbol{\alpha}$ was updated until the stationary condition of log-marginal w.r.t. $\boldsymbol{\alpha}$ approximately held.

In HCA with ARD, called HCA-ARD, the initial values of $U$ and $V$ were obtained by SVD. We also examined an ARD-based procedure with another initial value setting, i.e., starting from the result obtained by HCA-greedy, which is signified by HCA-g+ARD.

## 4  Related work

In this work, the ideas from both probabilistic modeling of linear component analyzers and sparse matrix factorization frameworks are combined into an analytical tool for data with underlying heterogeneous structures.

The weighted low-rank matrix factorization (WLRMF) [3] has been proposed as a minimization problem of the weighted error:

$$\min_{\boldsymbol{U}, \boldsymbol{V}} = \sum_{i,j} w_{ij} (y_{ij} - \sum_{k} u_{ik} v_{jk})^2, \tag{12}$$

where $w_{ij}$ is a weight for the element $y_{ij}$ of the observation matrix $\boldsymbol{Y}$. The weight value is set as $w_{ij} = 0$ if the corresponding $y_{ij}$ is missing or $w_{ij} > 0$ otherwise. This objective function is equivalent to the (negative) log-likelihood of a probabilistic generative model based on an assumption that each element of the residual matrix obeys a Gaussian distribution with variance $1/w_{ij}$. The WLRMF objective function is equivalent to our log-likelihood function (4) if the weight is set at estimated inverse noise variance for each $(i,j)$-th element. Although the prior term, $\ln p(\boldsymbol{V}) = -\frac{1}{2} \sum_{jk} v_{jk}^2 + \text{const.}$, has been added to eq. (4), it just imposes a constraint on the linear indeterminacy between $U$ and $V$, and hence the resultant low-rank matrix $\boldsymbol{U}\boldsymbol{V}^{\mathrm{T}}$ is identical to that by WLRMF.

Bayesian PCA [2] is also a matrix factorization procedure, which includes a characteristic prior density of factor-loading vectors, $\ln p(\boldsymbol{U}|\boldsymbol{\alpha}) = -\frac{1}{2} \sum_{ik} \alpha_k u_{ik}^2 + \text{const.}$. It is an equivalent prior for

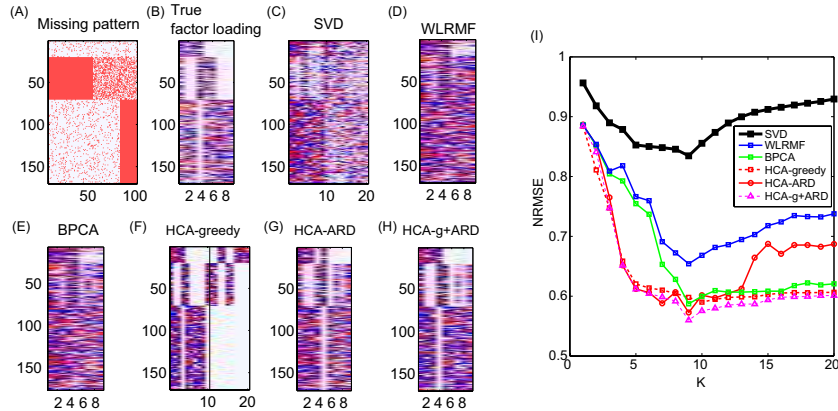

Figure 2: Experimental results when applied to an artificial data matrix. (A) Missing pattern of the observation matrix. Vertical and horizontal axes correspond to row (typically, genes) and column (typically, samples) of the matrix (typically, gene expression matrix). Red cells signify missing elements. (B) True factor-loading matrix. Horizontal axis denotes factors. Color and its intensity denote element values and white cells denote zero elements. Panels from (C) to (H) show the factor-loading matrices estimated by SVD, WLRMF, BPCA, HCA-greedy, HCA-ARD, and HCA-g+ARD, respectively. The vertical line in panel (F) denotes the automatically determined number of components. Panel (I) shows missing value prediction performance obtained by the three HCA algorithms and other methods. The vertical and horizontal axes denote normalized root mean square of test errors and dimensionalities of factors, respectively.

HCA-ARD (eq. (10)) if we assume only a single block. Although this prior term obviously a simple $L^2$ norm in the WLRMF, it also includes hyper parameter $\boldsymbol{\alpha}$ which constitute different regularization term and it leads to automatic model (intrinsic dimensionality) selection when $\boldsymbol{\alpha}$ is determined by evidence criterion.

Component analyzers with sparse factor-loadings have recently been investigated as sparse PCA (SPCA). In a well established context of SPCA studies (e.g. [4]), the tradeoff problem is solved between the understandability (sparsity of factor-loadings) and the reproducibility of the covariance matrix from the sparsified factor-loadings. In our HCA, the block-wise sparse factor-loading matrix is useful not only for understandability but also for generalization ability. The latter merit comes from the assumption that the observation includes uncertainty due to a small sample size, large noises, and missing observations, which have not been considered sufficiently in SPCA.

## 5 Experiments

**Experiment 1: an artificial dataset**   We prepared an artificial data set with an underlying block structure. For this we generated a $170 \times 9$ factor-loading matrix $\boldsymbol{U}$ that included a pre-determined block structure (white vs. colored in Fig. 2(B)), and a $100 \times 9$ factor matrix $\boldsymbol{V}$ by applying orthogonalization to the factors sampled from a standard Gaussian distribution. The observation matrix $\boldsymbol{Y}$ was produced by $\boldsymbol{U}\boldsymbol{V}^{\mathrm{T}} + \boldsymbol{E}$, where each element of $\boldsymbol{E}$ was generated from a standard Gaussian. Then, missing values were artificially introduced according to the pre-determined block structure (Fig. 2(A)).

- Block 1 consisted of 20 features with randomly selected 10 % missing entries.
- Block 2 consisted of 50 features whose 50% columns were completely missing and the remaining columns contained randomly selected 50% missing entries.
- Block 3 consisted of 100 features whose 20% columns were completely missing and the remaining columns contained randomly selected 20% missing entries.

We applied three HCA algorithms: HCA-greedy, HCA-ARD, and HCA-g+ARD, and three existing matrix factorization algorithms: SVD, WLRMF and BPCA.

**SVD** SVD calculated for a matrix whose missing values are imputed to zeros.

**WLRMF[3]** The weights were set 1 for the value-existing entries or 0 for the missing entries.

**BPCA** WLRMF with an ARD prior, called here BPCA, which is equivalent to HCA-ARD except that all features are in a single active block (i.e., colored in Fig. 2(B)). We confirmed this method exhibited almost the same performance as VB-EM-based algorithm [5].

The generalization ability was evaluated on the basis of the estimation performance for artificially introduced missing values. The estimated factor-loading matrices and missing value estimation accuracies are shown in Figure 2. Factor-loading matrices based on WLRMF and BPCA were obviously almost the same with that by SVD, because these three methods did not assume any sparsity in the factor-loading matrix.

The HCA-greedy algorithm terminated at $K = 10$. The factor-loading matrix estimated by HCA-greedy showed an identical sparse structure to the one consisting of the top five factors in the true factor-loadings. The sixth factor in the second block was not extracted, possibly because the second block lacked information due to the large rate of missing values. This algorithm also happened to extract one factor not included in the original factor-loadings, as the tenth one in the first block.

Although the HCA-ARD and HCA-g+ARD algorithms extracted good ones as the top three and four factors, respectively, they failed to completely reconstruct the sparsity structure in other factors. As shown in panel (I), however, such a poorly extracted structure did not increase the generalization error, implying that the essential structure underlying the data was extracted well by the three HCA-based algorithms.

Reconstruction of missing values was evaluated by normalized root mean square errors: NRMSE $\stackrel{\text{def}}{=}$ $\sqrt{\text{mean}[(y - \tilde{y})^2]/\text{var}[y]}$, where $y$ and $\tilde{y}$ denote true and estimated values, respectively, the mean is the average over all the missing entries and the variance is for all entries of the matrix.

Figure 2(I) shows the generalization ability of missing value predictions. SVD and WLRMF, which incurred no penalty on extracting a large number of factors, exhibited the best results around $K = 9$, but got worse with the increase in the number of $K$ due to over-fitting. HCA-g+ARD showed the best performance at $K = 9$, which was better than that obtained by all the other methods. HCA-greedy, HCA-ARD, and BPCA exhibited comparative performance at $K = 9$. At $K = 2, \ldots, 8$, the HCA algorithms performed better than BPCA. Namely, the sparse structure in the factor-loadings tended to achieve better performance. HCA-ARD performed less effectively than the other two HCA algorithms at $K > 13$, because of convergence to local solutions. This reason is supported by the fact that HCA-g+ARD employing good initialization by HCA-greedy exhibited the best performance among all the HCA algorithms. Accordingly, HCA showed a better generalization ability with a smaller number of effective parameters than the existing methods.

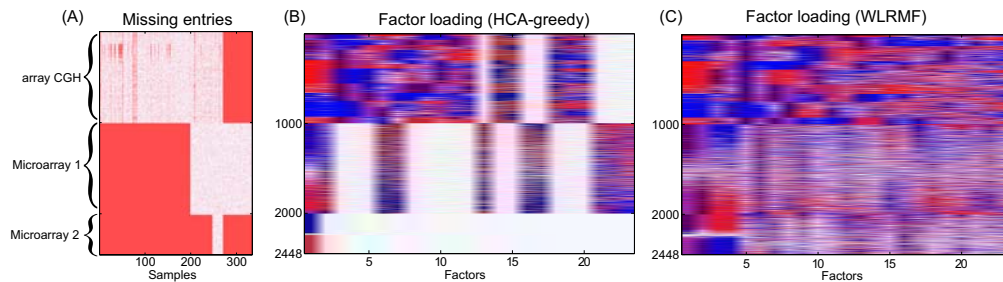

Figure 3: Analysis of an NBL dataset. Vertical axes denote high-dimensional features. Features measured by array CGH technology are sorted in the chromosomal order. Microarray features are sorted by correlations to sample's prognosis, dead or alive at the end of clinical followup. (A) Missing pattern in the NBL dataset. White and red colors denote observed and missing entries in the data matrix, respectively. (B) and (C) Factor-loading matrices estimated by the HCA-greedy and WLRMF algorithms, respectively.

**Experiment 2: a cross-analysis of neuroblastoma data** We next applied our HCA to a neuroblastoma (NBL) dataset consisting of three data blocks taken by three kinds of high-throughput genomic measurement technologies.

**Array CGH** Chromosomal changes of 2340 DNA segments (using 2340 probes) were measured for each of 230 NBL tumors, by using the array comparative genomic hybridization (array CGH) technology. Data for 1000 probes were arbitrarily selected from the whole dataset.

**Microarray 1** Expression levels of 5340 genes were measured for 136 tumors from NBL patients. We selected 1000 genes showing the largest variance over the 136 tumors.

**Microarray 2** Gene expression levels in 25 out of 136 tumors were also measured by a small-sized microarray technology harboring 448 probes.

The dataset Microarray 1 was the same one as used in the previous study [6], and the other two datasets, array CGH and Microarray 2, were also provided by the same research group for this study. As seen in Figure 3(A), the set of measured samples was quite different in the three experiments, leading to apparent block-wise missing observations. We normalized the data matrix so that the block-wise variances become unity. We further added 10% missing entries randomly into the observed entries in order to evaluate missing value prediction performance.

When HCA-greedy was applied to this dataset, it terminated at $K = 23$, but we continued to obtain further factors until $K = 80$. Figure 3(B) shows the factor-loading matrix from $K = 0$ to 23. HCA-greedy extracted one factor showing the relationship between the three measurement devices and three factors between aCGH and Microarray 1. The other factors accounted for either of aCGH or Microarray 1. The first factor was strongly correlated with patient's prognosis as clearly shown by the color code in the parts of Microarrays 1 and 2. Note that the features in these two datasets are aligned by correlations to the prognosis. This suggests that the dataset Microarray 2 did not include factors other than the first one as those strongly related to the prognosis. On the other hand, WLRMF extracted the identical first factor to HCA-greedy, but extracted much more factors concerning Microarray 2, all of which may not be trustworthy because the number of samples observed in Microarray 2 was as small as 25.

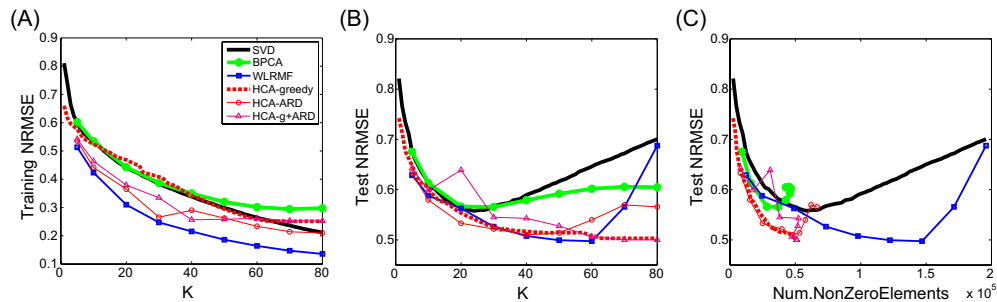

Figure 4: Missing value prediction performance by the six algorithms. Vertical axis denotes normalized root mean square of training errors (A) or test errors (B and C). Horizontal axis denotes the number of factors (A and B) or the number of non-zero elements in the factor-loading matrices (C). Each curve corresponds to one of the six algorithms.

We also applied SVD, WLRMF, BPCA and other two HCA algorithms to the NBL dataset. For WLRMF, BPCA, HCA-ARD, and HCA-g+ARD, the initial numbers of factors were set at $K = 5, 10, 20, \ldots, 70$, and 80. Missing value prediction performance in terms of NRMSE was obtained as a measurement value of generalization performance. Note that the original data matrix included many missing values, but we evaluated the performance by using artificially introduced missing values. Figure 4 shows the results.

Training errors almost monotonically decreased as the number of factors increased (Fig. 4A), indicating the stability of the algorithms. The only exception was HCA-ARD whose error increased from $K = 30$ to $K = 40$; this was due to local solution, because HCA-g+ARD employing the same algorithm but starting from different initialization showed consistent improvements in its performance.

Test errors did not show monotonic profiles except that HCA-greedy exhibited monotonically better results for larger $K$ values (Fig. 4B and C). SVD and WLRMF exhibited the best performance at $K = 22$ and $K = 60$, respectively, and got worse as the number of factors increased due to over-fitting.

Overall, the variants of our new HCA concept have shown good generalization performance as measured on missing values, much similar to existing methods like WLRMF. We would like to emphasize, however, that HCA yields a clearer *factor structure* that is easier interpretable from the biological point of view.

## 6    Conclusion

Complex structured data are ubiquitous in practice. For instance, when we should integrate data derived from different measurement devices, it becomes critically important to combine the information in each single source optimally — otherwise no gain can be achieved beyond the individual analyses. Our Bayesian HCA model allows to take into account such structured feature vectors that possess different intrinsic blocking characteristics. The new probabilistic structured matrix factorization framework was applied to toy data and to neuroblastoma data collected by multiple high-throughput measurement devices which had block-wise missing structures due to different experimental designs. HCA achieved a block-wise sparse factor-loading matrix, representing the information amount contained in each block of the dataset simultaneously. While HCA provided a better or similar missing value prediction performance than existing methods such as BPCA or WLRMF, the heterogeneous structure underlying the problem was clearly captured much better. Furthermore the HCA factors derived are an interesting representation that may ultimately lead to a better modeling of the neuroblastoma data (see section 5).
In the current HCA implementation, block structures were assumed to be known, as for the neuroblastoma data. Future work will go into a fully automatic estimate of structure from measured multi-modal data and the respective model selection techniques to achieve this goal.
Clearly there is an increasing need for methods that are able to reliably extract factors from multi-modal structured data with heterogeneous features. Our future effort will therefore strive towards applications beyond bioinformatics and to design novel structured spatio-temporal decomposition methods in applications like electroencephalography (EEG), image and audio analyses.

**Acknowledgement**    This work was supported by a Grant-in-Aid for Young Scientists (B) No. 19710172 from MEXT Japan.

## References

[1] I. Nabney and Christopher Bishop. Netlab: Netlab neural network software. http://www.ncrg.aston.ac.uk/netlab/, 1995.

[2] C.M. Bishop. Bayesian PCA. In *Proceedings of 11th conference on Advances in neural information processing systems*, pages 382–388. MIT Press Cambridge, MA, USA, 1999.

[3] N. Srebro and T. Jaakkola. Weighted low rank matrix approximations. In *Proceedings of 20th International Conference on Machine Learning*, pages 720–727, 2003.

[4] A. d'Aspremont, F. R. Bach, and L. El Ghaoui. Full regularization path for sparse principal component analysis. In *Proceedings of the 24th International Conference on Machine Learning*, 2007.

[5] S. Oba, M. Sato, I. Takemasa, M. Monden, K. Matsubara, and S. Ishii. A Bayesian missing value estimation method for gene expression profile data. *Bioinformatics*, 19(16):2088–2096, 2003.

[6] M. Ohira, S. Oba, Y. Nakamura, E. Isogai, S. Kaneko, A. Nakagawa, T. Hirata, H. Kubo, T. Goto, S. Yamada, Y. Yoshida, M. Fuchioka, S. Ishii, and A. Nakagawara. Expression profiling using a tumor-specific cDNA microarray predicts the prognosis of intermediate risk neuroblastomas. *Cancer Cell*, 7(4):337–350, Apr 2005.